# Novel iteration schemes for the Cluster Variation Method

**Hilbert J. Kappen**
Department of Biophysics
Nijmegen University
Nijmegen, the Netherlands
bert@mbfys.kun.nl

**Wim Wiegerinck**
Department of Biophysics
Nijmegen University
Nijmegen, the Netherlands
wimw@mbfys.kun.nl

## Abstract

The Cluster Variation method is a class of approximation methods containing the Bethe and Kikuchi approximations as special cases. We derive two novel iteration schemes for the Cluster Variation Method. One is a fixed point iteration scheme which gives a significant improvement over loopy BP, mean field and TAP methods on directed graphical models. The other is a gradient based method, that is guaranteed to converge and is shown to give useful results on random graphs with mild frustration. We conclude that the methods are of significant practical value for large inference problems.

## 1 Introduction

Belief Propagation (BP) is a message passing scheme, which is known to yield exact inference in tree structured graphical models [1]. It has been noted by several authors that Belief Propagation can can also give impressive results for graphs that are not trees [2].

The Cluster Variation Method (CVM), is a method that has been developed in the physics community for approximate inference in the Ising model [3]. The CVM approximates the joint probability distribution by a number of (overlapping) marginal distributions (clusters). The quality of the approximation is determined by the size and number of clusters. When the clusters consist of only two variables, the method is known as the Bethe approximation. Recently, the method has been introduced by Yedidia et al.[4] into the machine learning community, showing that in the Bethe approximation, the CVM solution coincides with the fixed points of the belief propagation algorithm. For clusters consisting of more than two variables, [4] present a message passing scheme called generalized belief propagation (GBP). This approximation to the free energy is often referred to as the Kikuchi approximation. They show, that GBP gives a significant improvement over the Bethe approximation for a small two dimensional Ising lattice with random couplings. However, for larger latices, both GBP and BP fail to converge [4, 5].

In [5] the CCCP method is proposed, which is a double loop iteration algorithm that is guaranteed to converge for the general CVM problem. Intuitively, the method

consists of iteration a sequence of convex subproblem (outer loop) each of which is solved using a fixed point iteration method (inner loop). In this sense, the method is similar to the UPS algorithm of [6] which identifies trees as subproblems.

In this paper, we propose two algorithms, one is a fixed point iteration procedure, the other a gradient based method. We show that the fixed point iteration method gives very fast convergence and accurate results for some classical directed graphical models. However, for more challenging cases the fixed point method does not converge and the gradient based approach, which is guaranteed to converge, is preferable.

## 2   The Cluster Variation Method

In this section, we briefly present the cluster variation method. For a more complete treatment see for instance [7]. Let $x = (x_1, \ldots, x_n)$ be a set of variables, where each $x_i$ can take a finite number of values. Consider a probability distribution on $x$ of the form

$$p_H(x) = \frac{1}{Z(H)} e^{-H(x)} \qquad Z = \sum_x e^{-H(x)}$$

It is well known, that $p_H$ can be obtained as the minimum of the free energy, which is a functional over probability distributions of the following form:

$$F_H(p) = \langle H \rangle + \langle \log p \rangle, \tag{1}$$

where the expectation value is taken with respect to the distribution $p$, i.e. $\langle H \rangle = \sum_x p(x) H(x)$. When one minimizes $F_H(p)$ with respect to $p$ under the constraint of normalization $\sum_x p(x) = 1$, one obtains $p_H$.

Computing marginals of $p_H$ such as $p_H(x_i)$ or $p_H(x_i, x_j)$ involves sums over all states, which is intractable for large $n$. Therefore, one needs tractable approximations to $p_H$. The cluster variation method replaces the probability distribution $p_H(x)$ by a large number of (possibly overlapping) probability distributions, each describing a sub set (cluster) of variables. Due to the one-to-one correspondence between a probability distribution and the minima of a free energy we can define approximate probability distributions by constructing approximate free energies and computing their minimum. This is achieved by approximating Eq. 1 in terms of the cluster probabilities. The solution is obtained by minimizing this approximate free energy subject to normalization and consistency constraints.

Define clusters as subsets of distinct variables: $x_\alpha = (x_{i_1}, \ldots, x_{i_k})$, with $1 \leq i_j \leq n$. Consider the set of clusters $P$ that describe the interactions in $H$ and write $H$ as a sum of these interactions:

$$H(x) = \sum_{\alpha \in P} H_\alpha^\dagger(x_\alpha)$$

We now define a set of clusters $B$, that will determine our approximation in the cluster variation method. For each cluster $\alpha \in B$, we introduce a probability distribution $p_\alpha(x_\alpha)$ which jointly must approximate $p(x)$. $B$ should at least contain the interactions in $p(x)$ in the following way: $\forall \alpha \in P \Rightarrow \exists \alpha' \in B, \alpha \subset \alpha'$. In addition, we demand that no two clusters in $B$ contain each other: $\alpha, \alpha' \in B \Rightarrow \alpha \not\subset \alpha', \alpha' \not\subset \alpha$. The minimal choice for $B$ is to chose clusters from $P$ itself. The maximal choice for $B$ is the cliques obtained when constructing the junction tree[8]. In this case, the clusters in $B$ form a tree structure and the CVM method is exact.

Define a set of clusters $M$ that consist of any intersection of a number of clusters of $B$: $M = \{\beta | \beta = \cap_k \alpha_k, \alpha_k \in B\}$, and define $U = B \cup M$. Once $U$ is given, we

define numbers $a_\beta$ recursively by the Moebius formula

$$1 = \sum_{\alpha \in U, \alpha \supset \beta} a_\alpha, \quad \forall \beta \in U$$

In particular, this shows that $a_\alpha = 1$ for $\alpha \in B$.

The Moebius formula allows us to rewrite $\langle H \rangle$ in terms of the cluster probabilities $p_\alpha$:

$$\langle H \rangle = \sum_{\alpha \in U} a_\alpha \sum_{x_\alpha} p_\alpha(x_\alpha) H_\alpha(x_\alpha), \tag{2}$$

with $H_\alpha(x_\alpha) = \sum_{\beta \in P, \beta \subset \alpha} H_\beta^\dagger(x_\beta)$. Since interactions $H_\beta^\dagger$ may appear in more than one $H_\alpha$, the constants $a_\alpha$ ensure that double counting is compensated for.

Whereas $\langle H \rangle$ can be written exactly in terms of $p_\alpha$, this is not the case for the entropy term in Eq. 1. The approach is to decompose the entropy of a cluster $\alpha$ in terms of 'connected entropies' in the following way: [1]

$$S_\alpha = -\sum_{x_\alpha} p_\alpha(x_\alpha) \log p_\alpha(x_\alpha) = \sum_{\beta \subset \alpha} S_\beta^\dagger. \tag{3}$$

where the sum over $\beta$ contains all subclusters of $\alpha$. Such a decomposition can be made for any cluster. In particular it can be made for the 'cluster' consisting of all variables, so that we obtain

$$S = -\sum_x p(x) \log p(x) = \sum_\beta S_\beta^\dagger. \tag{4}$$

The cluster variation method approximates the total entropy by restricting this latter sum to only clusters in $U$ and re-expressing $S_\beta^\dagger$ in terms of $S_\alpha$, using the Moebius formula and the definition Eq. 3.

$$S \approx \sum_{\beta \in U} S_\beta^\dagger = \sum_{\beta \in U} \sum_{\alpha \supset \beta} a_\alpha S_\beta^\dagger = \sum_{\alpha \in U} a_\alpha S_\alpha \tag{5}$$

Since $S_\alpha$ is a function of $p_\alpha$ (Eq. 3), we have expressed the entropy in terms of cluster probabilities $p_\alpha$.

The quality of this approximation is illustrated in Fig. 1 for the SK model. Note, that both the Bethe and Kikuchi approximation strongly deteriorate around $J = 1$, which is where the spin-glass phase starts. For $J < 1$, the Kikuchi approximation is superior to the Bethe approximation. Note, however, that this figure only illustrates the quality of the truncations in Eq. 5 assuming that the exact marginals are known. It does not say anything about the accuracy of the approximate marginals using the approximate free energy.

Substituting Eqs. 2 and 5 into the free energy Eq. 1 we obtain the approximate free energy of the Cluster Variation method. This free energy must be minimized subject to normalization constraints $\sum_{x_\alpha} p_\alpha(x_\alpha) = 1$ and consistency constraints

$$p_\alpha(x_\beta) = p_\beta(x_\beta), \quad \alpha, \beta \in U, \beta \subset \alpha. \tag{6}$$

with $p_\alpha(x_\beta) = \sum_{x_{\alpha \setminus \beta}} p_\alpha(x_\alpha)$.

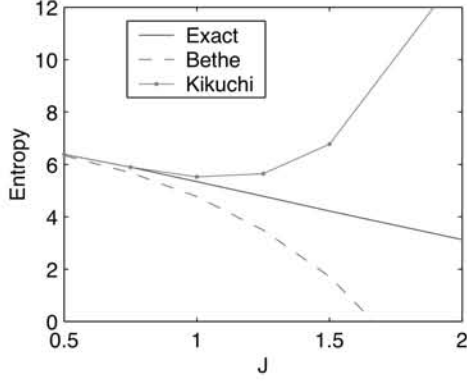

Figure 1: Exact and approximate entropies for the fully connected Boltzmann-Gibbs distribution on $n = 10$ variables with random couplings (SK model) as a function of mean coupling strength. Couplings $w_{ij}$ are chosen from a normal Gaussian distribution with mean zero and standard deviation $J/\sqrt{n}$. External fields $\theta_i$ are chosen from a normal Gaussian distribution with mean zero and standard deviation 0.1. The exact entropy is computed from Eq. 4. The Bethe and Kikuchi approximations are computed using the approximate entropy expression Eq. 5 with exact marginals and by choosing $B$ as the set of all pairs and all triplets, respectively.

The set of consistency constraints can be significantly reduced because some constraints imply others. Let $\alpha, \alpha', \dots$ denote clusters in $B$ and $\beta, \beta', \dots$ denote clusters in $M$.

- If $\beta \subset \beta' \subset \alpha$ and $p_\alpha(x_{\beta'}) = p_{\beta'}(x_{\beta'})$ and $p_\alpha(x_\beta) = p_\beta(x_\beta)$, then $p_{\beta'}(x_\beta) = p_\beta(x_\beta)$. This means that constraints between clusters in $M$ can be removed.

- If $\beta \subset \beta' \subset \alpha, \alpha'$ and $p_\alpha(x_{\beta'}) = p_{\alpha'}(x_{\beta'})$ and $p_\alpha(x_\beta) = p_\beta(x_\beta)$, then $p_{\alpha'}(x_\beta) = p_\beta(x_\beta)$. This means that some constraints between clusters in $B$ and $M$ can be removed.

We denote the remaining necessary constraints by $\alpha \to \beta$.

Adding Lagrange multipliers for the constraints we obtain the Cluster Variation free energy:

$$F_{\text{cvm}}(\{p_\alpha(x_\alpha)\}, \{\lambda_\alpha\}, \{\lambda_{\alpha\beta}(x_\beta)\}) = \sum_{\alpha \in U} a_\alpha \sum_{x_\alpha} p_\alpha(x_\alpha) \left(H_\alpha(x_\alpha) + \log p_\alpha(x_\alpha)\right)$$

$$- \sum_{\alpha \in U} \lambda_\alpha \left(\sum_{x_\alpha} p_\alpha(x_\alpha) - 1\right) - \sum_{\beta \in M} \sum_{\alpha \to \beta} \sum_{x_\beta} \lambda_{\alpha\beta}(x_\beta) \left(p_\alpha(x_\beta) - p_\beta(x_\beta)\right)$$

$$(7)$$

## 3 Iterating Lagrange multipliers

By setting $\frac{\partial F_{\text{cvm}}}{\partial p_\alpha(x_\alpha)}, \alpha \in U$ equal to zero, one can express the cluster probabilities in terms of the Lagrange multipliers:

$$p_\alpha(x_\alpha) = \frac{1}{Z_\alpha} \exp\left(-H_\alpha(x_\alpha) + \sum_{\beta \leftarrow \alpha} \lambda_{\alpha\beta}(x_\beta)\right) \tag{8}$$

$$p_\beta(x_\beta) = \frac{1}{Z_\beta} \exp\left(-H_\beta(x_\beta) - \frac{1}{a_\beta} \sum_{\alpha \to \beta} \lambda_{\alpha\beta}(x_\beta)\right) \tag{9}$$

The remaining task is to solve for the Lagrange multipliers such that all constraints (Eq. 6) are satisfied. We present two ways to do this.

When one substitutes Eqs. 8-9 into the constraint Eqs. 6 one obtains a system of coupled non-linear equations. In Yedidia et al.[4] a message passing algorithm was proposed to find a solution to this problem. Here, we will present an alternative method, that solves directly in terms of the Lagrange multipliers.

### 3.1 Fixed point iteration

Consider the constraints Eq. 6 for some fixed cluster $\beta$ and all clusters $\alpha \to \beta$ and define $B_\beta = \{\alpha \in B | \alpha \to \beta\}$. We wish to solve for all constraints $\alpha \to \beta$, with $\alpha \in B_\beta$ by adjusting $\lambda_{\alpha\beta}, \alpha \in B_\beta$. This is a sub-problem with $|B_\beta||x_\beta|$ equations and an equal number of unknowns, where $|B_\beta|$ is the number of elements of $B_\beta$ and $|x_\beta|$ is the number of values that $x_\beta$ can take. The probability distribution $p_\beta$ (Eq. 9) depends only on these Lagrange multipliers. $p_\alpha$ (Eq. 8) depends also on other Lagrange multipliers. However, we consider only its dependence on $\lambda_{\alpha\beta}, \alpha \in B_\beta$, and consider all other Lagrange multipliers as fixed. Thus,

$$p_\alpha(x_\alpha) = \exp(\lambda_{\alpha\beta}(x_\beta))\tilde{p}_\alpha(x_\alpha), \alpha \in B_\beta \tag{10}$$

with $\tilde{p}_\alpha$ independent of $\lambda_{\alpha\beta}, \alpha \in B_\beta$.

Substituting, Eqs. 9 and 10 into Eq. 6, we obtain a set of linear equations for $\lambda_{\alpha\beta}(x_\beta)$ which we can solve in closed form:

$$\lambda_{\alpha\beta}(x_\beta) = -\frac{a_\beta}{a_\beta + |B_\beta|} H_\beta(x_\beta) - \sum_{\alpha'} A_{\alpha\alpha'} \log \tilde{p}_{\alpha'}(x_\beta)$$

with

$$A_{\alpha\alpha'} = \delta_{\alpha\alpha'} - \frac{1}{a_\beta + |B_\beta|}$$

We update the probabilities with the new values of the Lagrange multipliers using Eqs. 9 and 10. We repeat the above procedure for all $\beta \in M$ until convergence.

### 3.2 Gradient descent

We define an auxiliary cost function

$$C = \sum_{\alpha\beta} \sum_{x_\beta} p_\beta(x_\beta) \log \frac{p_\beta(x_\beta)}{p_\alpha(x_\beta)} = \sum_{\alpha\beta} C_{\alpha\beta} \tag{11}$$

that is zero when all constraints are satisfied and positive otherwise and minimize this cost function with respect to the Lagrange multipliers $\lambda_{\alpha\beta}(x_\beta)$. The gradient

of $C$ is given by:

$$\frac{\partial C}{\partial \lambda_{\alpha\beta}(x_\beta)} = -\frac{p_\beta(x_\beta)}{a_\beta} \sum_{\alpha' \to \beta} (\log \frac{p_\beta(x_\beta)}{p_{\alpha'}(x_\beta)} - C_{\alpha'\beta}) - \sum_{\beta' \leftarrow \alpha} (p'_{\alpha\beta'}(x_\beta) - p_\alpha(x_\beta))$$

with

$$p'_{\alpha\beta}(x_\alpha) = p_\alpha(x_\alpha)\frac{p_\beta(x_\beta)}{p_\alpha(x_\beta)}$$

## 4 Numerical results

### 4.1 Directed Graphical models

We show the performance of the fixed point iteration procedure on several 'real world' directed graphical models. In figure 2a, we plot the exact single node marginals against the approximate marginals for the Asia problem [8]. Clusters in $B$ are defined according to the conditional probability tables. Convergence was reached in 6 iterations using fixed point iteration. Maximal error on the marginals is 0.0033. For comparison, we computed the mean field and TAP approximations, as previously introduced by [9]. Although TAP is significantly better than MF, it is far worse than the CVM method. This is not surprising, since both the MF and TAP approximation are based on single node approximation, whereas the CVM method uses potentials up to size 3.

In figure 2b, we plot the exact single node marginals against the approximate CVM marginals for the alarm network [10]. The structure and CPTs were downloaded from `www.cs.huji.ac.il/~nir`. Clusters in $B$ are defined according to the conditional probability tables and maximally contain 5 variables. Convergence was reached in 15 iterations using fixed point iteration. Maximal error on the marginals is 0.029. Ordinary loopy BP gives an error in the marginals of approximately 0.25 [2]. Mean field and TAP methods did not give reproducible results on this problem.

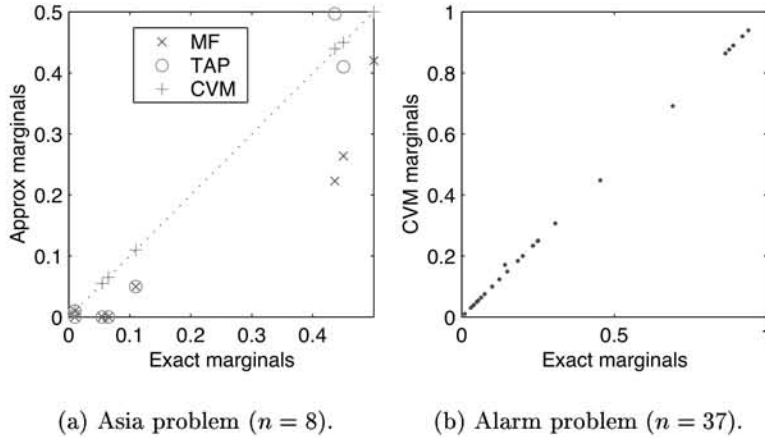

(a) Asia problem ($n = 8$).      (b) Alarm problem ($n = 37$).

Figure 2: Comparison of single node marginals on two real world problems.

Finally, we tested the cluster variation method on randomly generated directed

graphical models. Each node is randomly connected to $k$ parents. The entries of the probability tables are randomly generated between zero and one. Due to the large number of loops in the graph, the exact method requires exponential time in the maximum clique size, which can be seen from Table 1 to scale approximately linear with the network size. Therefore exact computation is only feasible for small graphs (up to size $n = 40$ in this case).

For the CVM, clusters in $B$ are defined according to the conditional probability tables. Therefore, maximal cluster size is $k + 1$. On these more challenging cases, the fixed point iteration method does not converge. The results shown are obtained with conjugate gradient descent on the auxiliary cost function Eq. 11. The results are shown in Table 1.

| $n$ | Iter | $|C|$ | Potential error | Margin error | $C$ |
|-----|------|-------|-----------------|--------------|-----|
| 10 | 16 | 8 | 0.018 | 0.004 | 9.7e-11 |
| 20 | 189 | 12 | 0.019 | 0.029 | 2.4e-4 |
| 30 | 157 | 16 | 0.033 | 0.130 | 2.1e-3 |
| 40 | 148 | 21 | 0.048 | 0.144 | 3.6e-3 |
| 50 | 132 | 26 | – | – | 4.5e-3 |

Table 1: Comparison of CVM method for large directed graphical models. Each node is connected to $k = 5$ parents. $|C|$ is the tree width of the triangulated graph required for the exact computation. Iter is the number of conjugate gradient descent iterations of the CVM method. Potential error and margin error are the maximum absolute distance (MAD) in any of the cluster probabilities and single variable marginals computed with CVM, respectively. $C$ is given by Eq. 11 after termination of CVM.

### 4.2 Markov networks

We compare the Bethe and Kikuchi approximations for the SK model with $n = 5$ neurons as defined in Fig. 1. We expect that for small $J$ the CVM approximation gives accurate results and deteriorates for larger $J$.

We compare the Bethe approximation, where we define clusters for all pairs of nodes and a Kikuchi approximation where we define clusters for all sub sets of three nodes. The results are given in Table 2. We see that for the Bethe approximation, the results of the fixed point iteration method (FPI) and the gradient based approach agree. For the Kikuchi approximation the fixed point iteration method does not converge and results are omitted. As expected, the Kikuchi approximation gives more accurate results than the Bethe approximation for small $J$.

## 5 Conclusion

We have presented two iteration schemes for finding the minimum of the constraint problem Eq. 7. One method is a fixed point iteration method that is equivalent to belief propagation for pairwise interactions. This method is very fast and gives very accurate results for 'not too complex' graphical models, such as real world directed graphical models and frustrated Boltzmann distributions in the Bethe approximation. However, for more complex graphs such as random directed graphs or more complex approximations, such as the Kikuchi approximation, the fixed point iteration method does not converge. Empirically, it is found that smoothing may somewhat help, but certainly does not solve this problem. For these more complex problems we propose to minimize an auxiliary cost function using a gradient

|  | Bethe | | | | Kikuchi | |
|---|---|---|---|---|---|---|
|  | FPI | | gradient | | gradient | |
| $J$ | Iter | Error | Iter | Error | Iter | Error |
| 0.25 | 7 | 0.000161 | 7 | 0.000548 | 120 | 0.000012 |
| 0.50 | 9 | 0.001297 | 11 | 0.001263 | 221 | 0.000355 |
| 0.75 | 13 | 0.004325 | 14 | 0.004392 | 86 | 0.021176 |
| 1.00 | 17 | 0.009765 | 15 | 0.009827 | 49 | 0.036882 |
| 1.50 | 38 | 0.027217 | 16 | 0.027323 | 150 | 0.059977 |
| 2.00 | 75 | 0.049955 | 20 | 0.049831 | 137 | 0.088481 |

Table 2: Comparison of Bethe and Kikuchi approximation for Boltzmann distributions. Iter is the number of iterations needed. Error is the MAD of single variable marginals.

based method. Clearly, this approach is guaranteed to converge. Empirically, we have found no problems with local minima. However, we have found that obtaining solution with $C$ close to zero may require many iterations.

**Acknowledgments**

This research was supported in part by the Dutch Technology Foundation (STW). I would like to thank Taylan Cemgil for providing his Matlab graphical models toolkit and Sebino Stramaglia (Bari, Italy) for useful discussions.

## Footnotes

[1] This decomposition is similar to writing a correlation in terms of means and covariance. For instance when $\alpha = (i)$, $S_{(i)} = S_{(i)}^\dagger$ is the usual mean field entropy and $S_{(ij)} = S_{(i)}^\dagger + S_{(j)}^\dagger + S_{(ij)}^\dagger$ defines the two node correction $S_{(ij)}^\dagger$.

# References

[1] J. Pearl. *Probabilistic reasoning in intelligent systems: Networks of Plausible Inference.* Morgan Kaufmann, San Francisco, California, 1988.

[2] Kevin P. Murphy, Yair Weiss, and Michael I. Jordan. Loopy belief propagation for approximate inference: An empirical study. In *Proceedings of Uncertainty in AI*, pages 467–475, 1999.

[3] R. Kikuchi. *Physical Review*, 81:988, 1951.

[4] J.S. Yedidia, W.T. Freeman, and Y. Weiss. Generalized belief propagation. In T.K. Leen, T.G. Dietterich, and V. Tresp, editors, *Advances in Neural Information Processing Systems 13 (Proceedings of the 2000 Conference)*, 2001. In press.

[5] A.L. Yuille and A. Rangarajan. The convex-concave principle. In T.G. Dieterich, S. Becker, and Z. Ghahramani, editors, *Advances in Neural Information Processing Systems*, volume 14, 2002. In press.

[6] Y. Teh and M. Welling. The unified propagation and scaling algorithm. In T.G. Dieterich, S. Becker, and Z. Ghahramani, editors, *Advances in Neural Information Processing Systems*, volume 14, 2002. In press.

[7] H.J. Kappen. The cluster variation method for approximate reasoning in medical diagnosis. In G. Nardulli and S. Stramaglia, editors, *Modeling Bio-medical signals*. World-Scientific, 2002. In press.

[8] S.L. Lauritzen and D.J. Spiegelhalter. Local computations with probabilties on graphical structures and their application to expert systems. *J. Royal Statistical society B*, 50:154–227, 1988.

[9] H.J. Kappen and W.A.J.J. Wiegerinck. Second order approximations for probability models. In Todd Leen, Tom Dietterich, Rich Caruana, and Virginia de Sa, editors, *Advances in Neural Information Processing Systems 13*, pages 238–244. MIT Press, 2001.

[10] I. Beinlich, G. Suermondt, R. Chaves, and G. Cooper. The alarm monitoring system: A case study with two probabilistic inference techniques for belief networks. In *2'nd European Conference on AI in Medicine*, 1989.
